# Combining Classifiers Using Correspondence Analysis

**Christopher J. Merz**
Dept. of Information and Computer Science
University of California, Irvine, CA 92697-3425 U.S.A.
cmerz@ics.uci.edu

**Category**: Algorithms and Architectures.

## Abstract

Several effective methods for improving the performance of a single learning algorithm have been developed recently. The general approach is to create a set of learned models by repeatedly applying the algorithm to different versions of the training data, and then combine the learned models' predictions according to a prescribed voting scheme. Little work has been done in combining the predictions of a collection of models generated by many learning algorithms having different representation and/or search strategies. This paper describes a method which uses the strategies of stacking and correspondence analysis to model the relationship between the learning examples and the way in which they are classified by a collection of learned models. A nearest neighbor method is then applied within the resulting representation to classify previously unseen examples. The new algorithm consistently performs as well or better than other combining techniques on a suite of data sets.

## 1   Introduction

Combining the predictions of a set of learned models[1] to improve classification and regression estimates has been an area of much research in machine learning and neural networks [Wolpert, 1992, Merz and Pazzani, 1997, Perrone, 1994, Breiman, 1996, Meir, 1995]. The challenge of this problem is to decide which models to rely on for prediction and how much weight to give each. The goal of combining learned models is to obtain a more accurate prediction than can be obtained from any single source alone.

Recently, several effective methods have been developed for improving the performance of a single learning algorithm by combining multiple learned models generated using the algorithm. Some examples include bagging [Breiman, 1996], boosting [Freund, 1995], and error correcting output codes [Kong and Dietterich, 1995]. The general approach is to use a particular learning algorithm and a model generation technique to create a set of learned models and then combine their predictions according to a prescribed voting scheme. The models are typically generated by varying the training data using resampling techniques such as bootstrapping [Efron and Tibshirani, 1993] or data partitioning [Meir, 1995]. Though these methods are effective, they are limited to a single learning algorithm by either their model generation technique or their method of combining.

Little work has been done in combining the predictions of a collection of models generated by many learning algorithms each having different representation and/or search strategies. Existing approaches typically place more emphasis on the model generation phase rather than the combining phase [Opitz and Shavlik, 1996]. As a result, the combining method is rather limited. The focus of this work is to present a more elaborate combining scheme, called SCANN, capable of handling any set of learned models, and evaluate it on some real-world data sets. A more detailed analytical and empirical study of the SCANN algorithm is presented in [Merz, 1997].

This paper describes a combining method applicable to model sets that are homogeneous or heterogeneous in their representation and/or search techniques. Section 2 describes the problem and explains some of the caveats of solving it. The SCANN algorithm (Section 3), uses the strategies of stacking [Wolpert, 1992] and correspondence analysis [Greenacre, 1984] to model the relationship between the learning examples and the way in which they are classified by a collection of learned models. A nearest neighbor method is then applied to the resulting representation to classify previously unseen examples.

In an empirical evaluation on a suite of data sets (Section 4), the naive approach of taking the plurality vote (PV) frequently exceeds the performance of the constituent learners. SCANN, in turn, matches or exceeds the performance of PV and several other stacking-based approaches. The analysis reveals that SCANN is not sensitive to having many poor constituent learned models, and it is not prone to overfit by reacting to insignificant fluctuations in the predictions of the learned models.

## 2   Problem Definition and Motivation

The problem of generating a set of learned models is defined as follows. Suppose two sets of data are given: a learning set $\mathcal{L} = \{(\mathbf{x}_i, y_i), i = 1, \ldots, I\}$ and a test set $\mathcal{T} = \{(\mathbf{x}_t, y_t), t = 1, \ldots, T\}$. $\mathbf{x}_i$ is a vector of input values which are either nominal or numeric values, and $y_i \in \{\mathcal{C}_1, \ldots, \mathcal{C}_C\}$ where $C$ is the number of classes. Now suppose $\mathcal{L}$ is used to build a set of $N$ functions, $\mathcal{F} = \{f_n(x)\}$, each element of which approximates $f(x)$, the underlying function.

The goal here is to combine the predictions of the members of $\mathcal{F}$ so as to find the best approximation of $f(x)$. Previous work [Perrone, 1994] has indicated that the ideal conditions for combining occur when the errors of the learned models are uncorrelated. The approaches taken thus far attempt to generate learned models which make uncorrelated errors by using the same algorithm and presenting different samples of the training data [Breiman, 1996, Meir, 1995], or by adjusting the search heuristic slightly [Opitz and Shavlik, 1996, Ali and Pazzani, 1996].

No single learning algorithm has the right bias for a broad selection of problems.

Therefore, another way to achieve diversity in the errors of the learned models generated is to use completely different learning algorithms which vary in their method of search and/or representation. The intuition is that the learned models generated would be more likely to make errors in different ways. Though it is not a requirement of the combining method described in the next section, the group of learning algorithms used to generate $\mathcal{F}$ will be heterogeneous in their search and/or representation methods (i.e., neural networks, decision lists, Bayesian classifiers, decision trees with and without pruning, etc.). In spite of efforts to diversify the errors committed, it is still likely that some of the errors will be correlated because the learning algorithms have the same goal of approximating $f$, and they may use similar search strategies and representations. A robust combining method must take this into consideration.

# 3 Approach

The approach taken consists of three major components: Stacking, Correspondence Analysis, and Nearest Neighbor (SCANN). Sections 3.1-3.3 give a detailed description of each component, and section 3.4 explains how they are integrated to form the SCANN algorithm.

## 3.1 Stacking

Once a diverse set of models has been generated, the issue of how to combine them arises. Wolpert [Wolpert, 1992] provided a general framework for doing so called *stacked generalization* or *stacking*. The goal of stacking is to combine the members of $\mathcal{F}$ based on information learned about their particular biases with respect to $\mathcal{L}^2$.

The basic premise of stacking is that this problem can be cast as another induction problem where the input space is the (approximated) outputs of the learned models, and the output space is the same as before, i.e.,

$$\mathcal{L}_1 = \{((\hat{f}_1(\mathbf{x}_i), \hat{f}_2(\mathbf{x}_i), \ldots, \hat{f}_N(\mathbf{x}_i)), y_i), i = 1, \ldots, I\}$$

The approximated outputs of each learned model, represented as $\hat{f}_n(\mathbf{x}_i)$, are generated using the following in-sample/out-of-sample approach:

1. Divide the $\mathcal{L}_0$ data up into $V$ partitions.
2. For each partition, $v$,
   - Train each algorithm on all but partition $v$ to get $\{\hat{f}_n^{-v}\}$.
   - Test each learned model in $\{\hat{f}_n^{-v}\}$ on partition $v$.
   - Pair the predictions on each example in partition $v$ (i.e., the new *input space*) with the corresponding output, and append the new examples to $\mathcal{L}_1$
3. Return $\mathcal{L}_1$

## 3.2 Correspondence Analysis

Correspondence Analysis (CA) [Greenacre, 1984] is a method for geometrically exploring the relationship between the rows and columns of a matrix whose entries are categorical. The goal here is to explore the relationship between the training

Table 1: Correspondence Analysis calculations.

| Stage | Symbol | Definition | Description |
|---|---|---|---|
| 1 | $\mathbf{N}$ | $(I \times J)$ indicator matrix | Records votes of learned models. |
| | $n$ | $\sum_{i=1}^{I} \sum_{j=1}^{J} n_{ij}$ | Grand total of table $\mathbf{N}$. |
| | $\mathbf{r}$ | $r_i = n_{i+}/n$ | Row masses. |
| | $\mathbf{c}$ | $c_j = n_{+j}/n$ | Column masses. |
| | $\mathbf{P}$ | $(1/n)\mathbf{N}$ | Correspondence matrix. |
| | $\mathbf{D_c}$ | $(J \times J)$ diagonal matrix | Masses $\mathbf{c}$ on diagonal. |
| | $\mathbf{D_r}$ | $(I \times I)$ diagonal matrix | Masses $\mathbf{r}$ on diagonal. |
| | $\mathbf{A}$ | $\mathbf{D_r}^{-1/2}(\mathbf{P} - \mathbf{rc}^T)\mathbf{D_c}^{-1/2}$ | Standardized residuals. |
| 2 | $\mathbf{A}$ | $\mathbf{U\Gamma V}^T$ | SVD of $\mathbf{A}$. |
| 3 | $\mathbf{F}$ | $\mathbf{D_r}^{-1/2}\mathbf{U\Gamma}$ | Principal coordinates of rows. |
| | $\mathbf{G}$ | $\mathbf{D_c}^{-1/2}\mathbf{V\Gamma}$ | Principal coordinates of columns. |

examples and how they are classified by the learned models. To do this, the prediction matrix, $\mathbf{M}$, is explored where $m_{in} = \hat{f}_n(\mathbf{x}_i)$ $(1 \leq i \leq I$, and $1 \leq n \leq N)$. It is also important to see how the predictions for the training examples relate to their true class labels, so the class labels are appended to form $\mathbf{M}'$, an $(I \times J)$ matrix (where $J = N+1$). For proper application of correspondence analysis, $\mathbf{M}'$ must be converted to an $(I \times (J \cdot C))$ *indicator matrix*, $\mathbf{N}$, where $n_{i,(j \cdot J+c)}$ is a one exactly when $m_{ij} = \mathcal{C}_c$, and zero otherwise.

The calculations of CA may be broken down into three stages (see Table 1). Stage one consists of some preprocessing calculations performed on $\mathbf{N}$ which lead to the *standardized residual matrix*, $\mathbf{A}$. In the second stage, a singular value decomposition (SVD) is performed on $\mathbf{A}$ to redefine it in terms of three matrices: $\mathbf{U}_{(I \times K)}$, $\mathbf{\Gamma}_{(K \times K)}$, and $\mathbf{V}_{(K \times J)}$, where $K = min(I - 1, J - 1)$. These matrices are used in the third stage to determine $\mathbf{F}_{(I \times K)}$ and $\mathbf{G}_{(J \times K)}$, the coordinates of the rows and columns of $\mathbf{N}$, respectively, in the new space. It should be noted that not all $K$ dimensions are necessary. Section 3.4, describes how the final number of dimensions, $K*$, is determined.

Intuitively, in the new geometric representation, two rows, $\mathbf{f}_{p*}$ and $\mathbf{f}_{q*}$, will lie close to one another when examples $p$ and $q$ receive similar predictions from the collection of learned models. Likewise, rows $\mathbf{g}_{r*}$ and $\mathbf{g}_{s*}$ will lie close to to one another when the learned models corresponding to $r$ and $s$ make similar predictions for the set of examples. Finally, each column, $r$, has a learned model, $j'$, and a class label, $c'$, with which it is associated; $\mathbf{f}_{p*}$ will lie closer to $\mathbf{g}_{r*}$ when model $j'$ predicts class $c'$.

### 3.3 Nearest Neighbor

The nearest neighbor algorithm is used to classify points in a weighted Euclidean space. In this scenario, each possible class will be assigned coordinates in the space derived by correspondence analysis. Unclassified examples will be mapped into the new space (as described below), and the class label corresponding to the closest class point is assigned to the example.

Since the actual class assignments for each example reside in the last $C$ columns of $\mathbf{N}$, their coordinates in the new space can be found by looking in the last $C$ *rows* of $\mathbf{G}$. For convenience, these class points will be called $Class_1, \ldots, Class_C$.

To classify an unseen example, $\mathbf{x}_{Test}$, the predictions of the learned models on $\mathbf{x}_{Test}$ must be converted to a *row profile*, $\tilde{\mathbf{r}}^T$, of length $J \cdot C$, where $\tilde{r}^T_{(j \cdot J+c)}$ is $1/J$ exactly

Table 2: Experimental results.

| Data set | PV error | SCANN vs PV ratio | S-BP vs PV ratio | S-BAYES vs PV ratio | Best Ind. vs PV ratio |
|---|---|---|---|---|---|
| abalone | 80.35 | **.490** | **.499** | **.487** | $.535^{BP}$ |
| bal | 13.81 | **.900** | **.859** | .992 | $.911^{BP}$ |
| breast | 4.31 | **.886** | .920 | **.881** | $.938^{BP}$ |
| credit | 13.99 | .999 | 1.012 | 1.001 | $1.054^{BP}$ |
| dementia | 32.78 | .989 | 1.037 | .932 | $1.048^{C4.5}$ |
| glass | 31.44 | 1.008 | **1.158** | **1.215** | $1.155^{OC1}$ |
| heart | 18.17 | .964 | .998 | .972 | $.962^{BP}$ |
| ionosphere | 3.05 | **.691** | **1.289** | **1.299** | $2.175^{C4.5}$ |
| iris | 4.44 | 1.017 | 1.033 | **1.467** | $1.150^{OC1}$ |
| krk | 6.67 | 1.030 | 1.080 | **1.149** | $1.159^{NN}$ |
| liver | 29.33 | 1.035 | 1.077 | **1.024** | $1.138^{CN2}$ |
| lymphography | 17.78 | 1.017 | **1.162** | 1.100 | $.983^{Pebls}$ |
| musk | 13.51 | **.812** | **.889** | **.835** | $1.113^{Pebls}$ |
| retardation | 32.64 | **.970** | **.960** | **.990** | $.936^{Bayes}$ |
| sonar | 23.02 | .990 | 1.079 | 1.007 | $1.048^{BP}$ |
| vote | 5.24 | **.903** | .908 | **.893** | $.927^{C4.5}$ |
| wave | 21.94 | 1.008 | **1.109** | 1.008 | $1.200^{Pebls}$ |
| wdbc | 4.27 | 1.000 | 1.103 | 1.007 | $1.164^{NN}$ |

when $m_{ij} = C_c$, and zero otherwise. However, since the example is unclassified, $\mathbf{x}_{Test}$ is of length $(J-1)$ and can only be used to fill the first $((J-1) \cdot C)$ entries in $\tilde{\mathbf{r}}^T$. For this reason, $C$ different versions are generated, i.e., $\tilde{\mathbf{r}}_1^T, \ldots, \tilde{\mathbf{r}}_C^T$, where each one "hypothesizes" that $\mathbf{x}_{Test}$ belongs to one of the $C$ classes (by putting $1/J$ in the appropriate column). Locating these profiles in the scaled space is a matter of simple matrix multiplication, i.e., $\mathbf{f}_c^T = \tilde{\mathbf{r}}_c^T \mathbf{G}\mathbf{\Gamma}^{-1}$. The $\mathbf{f}_c^T$ which lies closest to a class point, say $Class_{c'}$, is considered the "correct" hypothesized class, and $\mathbf{x}_{Test}$ is assigned the class label $c'$.

## 3.4 The SCANN Algorithm

Now that the three main parts of the approach have been described, a summary of the SCANN algorithm can be given as a function of $\mathcal{L}_0$ and the constituent learning algorithms, $\mathcal{A}$. The first step is to use $\mathcal{L}_0$ and $\mathcal{A}$ to generate the stacking data, $\mathcal{L}_1$, capturing the approximated predictions of each learned model. Next, $\mathcal{L}_1$ is used to form the indicator matrix, $\mathbf{N}$. A correspondence analysis is performed on $\mathbf{N}$ to derive the scaled space, $\mathbf{A} = \mathbf{U}\mathbf{\Gamma}\mathbf{V}^T$. The number of dimensions retained from this new representation, $K*$, is the value which optimizes classification on $\mathcal{L}_1$. The resulting scaled space is used to derive the row/column coordinates $\mathbf{F}$ and $\mathbf{G}$, thus geometrically capturing the relationships between the examples, the way in which they are classified, and their position relative to the true class labels. Finally, the nearest neighbor strategy exploits the new representation by predicting which class is most likely according to the predictions made on a novel example.

## 4   Experimental Results

The constituent learning algorithms, $\mathcal{A}$, spanned a variety of search and/or representation techniques: Backpropagation (BP) [Rumelhart et al., 1986], CN2 [Clark and Niblett, 1989], C4.5 [Quinlan, 1993], OC1 [Salzberg; and Beigel, 1993], PEBLS [Cost, 1993], nearest neighbor (NN), and naive Bayes. Depending on the data set, anywhere from five to eight instantiations of algorithms were applied. The combining strategies evaluated were PV, SCANN, and two other learners trained on $\mathcal{L}_1$: S-BP, and S-Bayes.

The data sets used were taken from the UCI Machine Learning Database Repository [Merz and Murphy, 1996], except for the unreleased medical data sets: *retardation* and *dementia*. Thirty runs per data set were conducted using a training/test partition of 70/30 percent. The results are reported in Table 2. The first column gives the mean error rate over the 30 runs of the baseline combiner, PV. The next three columns ("SCANN vs PV", "S-BP vs PV", and "S-Bayes vs PV") report the ratio of the other combining strategies to the error rate of PV. The column labeled "Best Ind. vs PV" reports the ratio with respect to the model with the best average error rate. The superscript of each entry in this column denotes the winning algorithm. A value less than 1 in the "*a* vs *b*" columns represents an improvement by method *a* over method *b*. Ratios reported in **boldface** indicate the difference between method *a* and method *b* is significant at a level better than 1 percent using a two-tailed sign test.

It is clear that, over the 18 data sets, SCANN holds a statistically significant advantage on 7 sets improving upon PV's classification error by 3-50 percent. Unlike the other combiners, SCANN posts no statistically significant losses to PV (i.e., there were 4 losses each for S-BP and S-Bayes). With the exception of the *retardation* data set, SCANN consistently performs as well or better than the best individual learned model. In the direct comparison of SCANN with the S-BP and S-Bayes, SCANN posts 5 and 4 significant wins, respectively, and no losses.

The most dramatic improvement of the combiners over PV came in the *abalone* data set. A closer look at the results revealed that 7 of the 8 learned models were very poor classifiers with error rates around 80 percent, and the errors of the poor models were highly correlated. This empirically demonstrates PV's known sensitivity to learned models with highly correlated errors. On the other hand, PV performs well on the *glass* and *wave* data sets where the errors of the learned models are measured to be fairly uncorrelated. Here, SCANN performs similarly to PV, but S-BP and S-Bayes appear to be overfitting by making erroneous predictions based on insignificant variations on the predictions of the learned models.

## 5   Conclusion

A novel method has been introduced for combining the predictions of heterogeneous or homogeneous classifiers. It draws upon the methods of stacking, correspondence analysis and nearest neighbor. In an empirical analysis, the method proves to be insensitive to poor learned models and matches the performance of plurality voting as the errors of the learned models become less correlated.

## Footnotes

[1]A learned model may be anything from a decision/regression tree to a neural network.

[2]Henceforth $\mathcal{L}$ will be referred to as $\mathcal{L}_0$ for clarity.

## References

[Ali and Pazzani, 1996] Ali, K. and Pazzani, M. (1996). Error reduction through learning multiple descriptions. *Machine Learning*, 24:173.

[Breiman, 1996] Breiman, L. (1996). Bagging predictors. *Machine Learning*, 24(2):123-40.

[Clark and Niblett, 1989] Clark, P. and Niblett, T. (1989). The CN2 induction algorithm. *Machine Learning*, 3(4):261-283.

[Cost, 1993] Cost, S.; Salzberg, S. (1993). A weighted nearest neighbor algorithm for learning with symbolic features. *Machine Learning*, 10(1):57-78.

[Efron and Tibshirani, 1993] Efron, B. and Tibshirani, R. (1993). *An Introduction to the Bootstrap*. Chapman and Hall, London and New York.

[Freund, 1995] Freund, Y. (1995). Boosting a weak learning algorithm by majority. *Information and Computation*, 121(2):256-285. Also appeared in COLT90.

[Greenacre, 1984] Greenacre, M. J. (1984). *Theory and Application of Correspondence Analysis*. Academic Press, London.

[Kong and Dietterich, 1995] Kong, E. B. and Dietterich, T. G. (1995). Error-correcting output coding corrects bias and variance. In *Proceedings of the 12th International Conference on Machine Learning*, pages 313-321. Morgan Kaufmann.

[Meir, 1995] Meir, R. (1995). Bias, variance and the combination of least squares estimators. In Tesauro, G., Touretzky, D., and Leen, T., editors, *Advances in Neural Information Processing Systems*, volume 7, pages 295-302. The MIT Press.

[Merz, 1997] Merz, C. (1997). Using correspondence analysis to combine classifiers. *Submitted to Machine Learning*.

[Merz and Murphy, 1996] Merz, C. and Murphy, P. (1996). UCI repository of machine learning databases.

[Merz and Pazzani, 1997] Merz, C. J. and Pazzani, M. J. (1997). Combining neural network regression estimates with regularized linear weights. In Mozer, M., Jordan, M., and Petsche, T., editors, *Advances in Neural Information Processing Systems*, volume 9. The MIT Press.

[Opitz and Shavlik, 1996] Opitz, D. W. and Shavlik, J. W. (1996). Generating accurate and diverse members of a neural-network ensemble. In Touretzky, D. S., Mozer, M. C., and Hasselmo, M. E., editors, *Advances in Neural Information Processing Systems*, volume 8, pages 535-541. The MIT Press.

[Perrone, 1994] Perrone, M. P. (1994). Putting it all together: Methods for combining neural networks. In Cowan, J. D., Tesauro, G., and Alspector, J., editors, *Advances in Neural Information Processing Systems*, volume 6, pages 1188-1189. Morgan Kaufmann Publishers, Inc.

[Quinlan, 1993] Quinlan, R. (1993). *C4.5 Programs for Machine Learning*. Morgan Kaufmann, San Mateo, CA.

[Rumelhart et al., 1986] Rumelhart, D. E., Hinton, G. E., and Williams, R. J. (1986). Learning internal representations by error propagation. In Rumelhart, D. E., McClelland, J. L., and the PDP research group., editors, *Parallel distributed processing: Explorations in the microstructure of cognition, Volume 1: Foundations*. MIT Press.

[Salzberg; and Beigel, 1993] Salzberg;, S. M. S. K. S. and Beigel, R. (1993). OC1: Randomized induction of oblique decision trees. In *Proceedings of AAAI-93*. AAAI Pres.

[Wolpert, 1992] Wolpert, D. H. (1992). Stacked generalization. *Neural Networks*, 5:241-259.